# MAP Inference in Chains using Column Generation

**David Belanger,**\* **Alexandre Passos,**\* **Sebastian Riedel**†**, Andrew McCallum**
Department of Computer Science, University of Massachusetts, Amherst
† Department of Computer Science, University College London
{belanger,apassos,mccallum}@cs.umass.edu, s.riedel@cs.ucl.ac.uk

## Abstract

Linear chains and trees are basic building blocks in many applications of graphical models, and they admit simple exact maximum a-posteriori (MAP) inference algorithms based on message passing. However, in many cases this computation is prohibitively expensive, due to quadratic dependence on variables' domain sizes. The standard algorithms are inefficient because they compute scores for hypotheses for which there is strong negative local evidence. For this reason there has been significant previous interest in beam search and its variants; however, these methods provide only approximate results. This paper presents new *exact* inference algorithms based on the combination of *column generation* and pre-computed bounds on terms of the model's scoring function. While we do not improve worst-case performance, our method substantially speeds real-world, typical-case inference in chains and trees. Experiments show our method to be twice as fast as exact Viterbi for Wall Street Journal part-of-speech tagging and over thirteen times faster for a joint part-of-speed and named-entity-recognition task. Our algorithm is also extendable to new techniques for approximate inference, to faster 0/1 loss oracles, and new opportunities for connections between inference and learning. We encourage further exploration of high-level reasoning about the optimization problem implicit in dynamic programs.

## 1 Introduction

Many uses of graphical models either directly employ chains or tree structures—as in part-of-speech tagging—or employ them to enable inference in more complex models—as in junction trees and tree block coordinate descent [1]. Traditional message-passing inference in these structures requires an amount of computation dependent on the product of the domain sizes of variables sharing an edge in the graph. Even in chains, exact inference is prohibitive in tasks with large domains due to the quadratic dependence on domain size. For this reason, many practitioners rely on beam search or other approximate inference techniques [2]. However, inference by beam search is approximate. This not only hurts test-time accuracy, but can also interfere with parameter estimation [3].

We present a new algorithm for exact MAP inference in chains that is substantially faster than Viterbi in the typical case. We draw on four key ideas: (1) it is wasteful to compute and store messages to and from low-scoring states, (2) it is possible to compute bounds on data-independent (not varying with the input data) scores of the model offline, (3) inference should make decisions based on local evidence for variables' values and rely on the graph only for disambiguation [4], and (4) runtime behavior should adapt to the cost structure of the model (i.e., the algorithm should be energy-aware [5]). The combination of these ideas yields provably exact MAP inference for chains and trees that can be more than an order of magnitude faster than traditional methods. Our algorithm has wide-ranging applicability, and we believe it could beneficially replace many traditional uses of Viterbi and beam search.

We exploit the connections between message-passing algorithms and LP relaxations for MAP inference. Directly solving LP relaxations for MAP using a state-of-the-art solver is inefficient because it ignores key structure of the problem [6]. However, it is possible to leverage message-passing as a fast subroutine to solve smaller LPs, and use high-level techniques to compose these solutions into a solution to the original problem.

With this interplay in mind, we employ column generation [7], a family of approaches to solving linear programs that are dual to cutting planes: they start by solving restricted primal problems, where many LP variables are set to zero, and slowly add other LP variables until they are able to prove that adding no other variable can improve the solution. From these properties of column generation, we also show how to perform approximate inference that is guaranteed not to be worse than the optimal by a given gap, how to construct an efficient 0/1-loss oracle by running 2-best inference in a subset of the graphical model, and how to learn parameters in such a way to make inference even faster.

The use of column generation has not been widely explored or appreciated in graphical models. This paper is intended to demonstrate its benefits and encourage further work in this direction. We demonstrate experimentally that our method has substantial speed advantages while retaining guaranteed exact inference. In Wall Street Journal part-of-speech tagging our method is more than 2.5 times faster than Viterbi, and also faster than beam search with a width of two. In joint POS tagging and named entity recognition, our method is thirteen times faster than Viterbi and also faster than beam search with a width of seven.

## 2   Delayed Column Generation in LPs

In LPs used for combinatorial optimization problems, we know a priori that there are optimal solutions in which many variables will be set to zero. This is enforced by the problem's constraints or it characterizes optimality (e.g., the solution to a shortest path LP would not include multiple paths). *Column generation* is a technique for exploiting this sparsity for faster inference. It restricts an LP to a subset of its variables (implicitly setting the others to zero) and alternates between solving this *restricted linear program* and selecting which variables should be added to it, based on whether they could potentially improve the objective. When no candidates remain, the current solution to the restricted problem is guaranteed to be the exact solution of the unrestricted problem.

The test to determine whether un-generated variables could potentially improve the objective is whether their *reduced cost* is positive, which is also the test employed by some pivoting rules in the simplex algorithm [8, 7]. The difference between the algorithms is that simplex enumerates primal variables explicitly, while column generation "generates" them only as needed. The key to an efficient column generation algorithm is an oracle that can either prove that no variable with positive reduced cost exists or produce one.

Consider the general LP:

$$\textbf{max.} \quad c^T x \quad \textbf{s.t.} \quad Ax \leq b, \quad x \geq 0 \tag{1}$$

With corresponding Lagrangian:

$$L(x, \lambda) = c^T x + \lambda^t (b - Ax) = \Sigma_i \left( c_i - A_i^T \lambda \right) x_i + \lambda^t b. \tag{2}$$

For a given assignment to the dual variables $\lambda$, a variable $x_i$ is a candidate for being added to the restricted problem if its *reduced cost* $r_i = c_i - A_i^T \lambda$, the scalar multiplying it in the Lagrangian, is positive. Another way to justify this decision rule is by considering the constraints in the LP dual:

$$\textbf{min.} \quad b^T \lambda \quad \textbf{s.t.} \quad A^T \lambda \geq c \quad \lambda \geq 0 \tag{3}$$

Here, the reduced cost of a primal variable equals the degree to which its dual constraint is violated, and thus column generation in the primal is equivalent to cutting planes in the dual [7]. If there is no variable of positive reduced cost, then the current dual variables from the restricted problem are feasible in the unrestricted problem, and thus we have a primal-dual optimal pair, and can terminate column generation. An advantageous property of column generation that we employ later on is that it maintains primal feasibility across iterations, and thus it can be halted to provide approximate, *anytime* solutions.

# 3 Connection Between LP Relaxations and Message-Passing in Chains

This section provides background showing how the LP formulation of the inference problem in chains leads to the known message-passing algorithm. The derivation follows Wainwright and Jordan [9], but is specialized for chains and highlights connections to our contributions.

The LP for MAP inference in chains is as follows

$$
\begin{aligned}
\textbf{max.} \quad & \sum_{i,x_i} \mu_i(x_i)\theta_i(x_i) + \sum_{i,x_i,x_{i+1}} \mu_i(x_i,x_{i+1})\tau(x_i,x_{i+1}) \\
\textbf{s.t.} \quad & \sum_{x_i} \mu_i(x_i) = 1 & \forall i \\
& \sum_{x_i} \mu_i(x_i,x_{i+1}) = \mu_{i+1}(x_{i+1}) & \forall i, x_{i+1} \\
& \sum_{x_{i+1}} \mu_i(x_i,x_{i+1}) = \mu_i(x_i) & \forall i, x_i
\end{aligned}
\tag{4}
$$

where $\theta_i(x_i)$ is the score obtained from assigning the $i$-th variable to value $x_i$, $\mu_i(x_i)$ is an indicator variable saying whether or not the MAP assignment sets the $i$-th variable to the value $x_i$, and $\tau(x_i, x_{i+1})$ is the score the model assigns to a transition from value $x_i$ to value $x_{i+1}$. It's implicitly assumed that all variables are positive. We assume a static $\tau$, but all statements trivially generalize to position-dependent $\tau_i$.

We can restructure this LP to only depend on the pairwise assignment variables $\mu_i(x_i, x_{i+1})$ by creating an edge between the last variable in the chain and an artificial variable and then "billing" all local scores to the pairwise edge that touches them from the right. Then we restructure the constraints to sum out both sides of each edge, and add indicator variables $\mu_n(x_n, \cdot)$ and 0-scoring transitions for the artificial edge. This leaves the following LP (with dual variables written after their corresponding constraints).

$$
\begin{aligned}
\textbf{max.} \quad & \sum_{i,x_i,x_{i+1}} \mu_i(x_i,x_{i+1})(\tau_i(x_i,x_{i+1}) + \theta_i(x_i)) \\
\textbf{s.t.} \quad & \sum_{x_n} \mu_n(x_n, \cdot) = 1 & (N) \\
& \sum_{x_{i-1}} \mu_{i-1}(x_{i-1},x_i) = \sum_{x_{i+1}} \mu_i(x_i,x_{i+1}) & (\alpha_i(x_i))
\end{aligned}
\tag{5}
$$

The dual of this linear program is

$$
\begin{aligned}
\textbf{min.} \quad & N \\
\textbf{s.t.} \quad & \alpha_{i+1}(x_{i+1}) - \alpha_i(x_i) \geq \tau(x_i,x_{i+1}) + \theta_i(x_i) & \forall i, x_i, x_{i+1} \\
& N - \alpha_n(x_n) \geq \theta_n(x_n) & \forall x_n
\end{aligned}
\tag{6}
$$

and setting the $\alpha$ dual variables by

$$
\alpha_{i+1}(x_{i+1}) = \max_{x_i} \alpha_i(x_i) + \theta_i(x_i) + \tau(x_i,x_{i+1})
\tag{7}
$$

and $N = \max_{x_n} \alpha_n(x_n) + \theta_n(x_n)$ is a sufficient condition for dual feasibility, and as $N$ will have the value of the primal solution, for optimality. Note that this equation is exactly the forward message-passing equation for max-product belief propagation in chains, i.e. the Viterbi algorithm.

A setting of the dual variables is optimal if maximization of the problem's Lagrangian over the primal variables yields a primal-feasible setting. The coefficients on the edge variables $\mu_i(x_i, x_{i+1})$ are their reduced costs,

$$
\alpha_i(x_i) - \alpha_{i+1}(x_{i+1}) + \theta_i(x_i) + \tau(x_i,x_{i+1}).
\tag{8}
$$

For duals that obey the constraints of (6), it is clear that the maximal reduced cost is zero, when $x_i$ is set to the argmax used when constructing $\alpha_{i+1}(x_{i+1})$. Therefore, to a obtain a primal-optimal solution, we start at the end of the chain and follow the argmax indices to the beginning, which is the same backward sweep of the Viterbi algorithm.

## 3.1 Improving the reduced cost with information from both ends of the chain

Column generation adds all variables with positive reduced cost to the restricted LP, but equation (8) leads to an inefficient algorithm because it is positive for many irrelevant edge settings. In (8), the only terms that involve $x_{i+1}$ are $\tau(x_i, x_{i+1})$ and the $\tau(x_i', x_{i+1})$ term that is part of $\alpha_{i+1}(x_{i+1})$. These are data-independent. Therefore, even if there is very strong local evidence against a particular

setting $x_{i+1}$, pairs $x_i, x_{i+1}$ may have positive reduced cost if the global transition factor $\tau(x_i, x_{i+1})$ places positive weight on their compatibility.

We can improve upon this by exploring different LP formulations than that of Wainwright and Jordan. Note that in equation (5) a local score is "billed" to its rightmost edge. Instead, if we split it halfway (now using phantom edges in both sides of the chain), we would obtain slightly different message passing rules and the following reduced cost expression:

$$\alpha_i(x_i) - \alpha_{i+1}(x_{i+1}) + \frac{1}{2}\left(\theta_i(x_i) + \theta_j(x_j)\right) + \tau(x_i, x_{i+1}). \tag{9}$$

This contains local information for both $x_i$ and $x_{i+1}$, though it halves the magnitude of it. In table 2 we demonstrate that this yields comparable performance to using the reduced cost of (8), which still outperforms Viterbi. An even better reduced cost expression can be obtained by duplicating the marginalization constraints, we have:

$$
\begin{aligned}
\textbf{max.} \quad & \sum_{i,x_i,x_{i+1}} \mu_i(x_i, x_{i+1})\left(\tau(x_i, x_{i+1}) + \tfrac{1}{2}\theta_i(x_i) + \tfrac{1}{2}\theta_{i+1}(x_{i+1})\right) \\
\textbf{s.t.} \quad & \sum_{x_n} \mu_n(x_n, \cdot) = 1 && (N^+) \\
& \sum_{x_1} \mu_0(\cdot, x_1) = 1 && (N^-) \\
& \sum_{x_{i-1}} \mu_{i-1}(x_{i-1}, x_i) = \sum_{x_{i+1}} \mu_i(x_i, x_{i+1}) && (\alpha_i(x_i)) \\
& \sum_{x_{i+1}} \mu_i(x_i, x_{i+1}) = \sum_{x_{i-1}} \mu_{i-1}(x_{i-1}, x_i) && (\beta_i(x_i))
\end{aligned}
\tag{10}
$$

Following similar logic as in the previous section, setting the dual variables according to (7) and

$$\beta_{i-1}(x_{i-1}) = \max_{x_i} \beta_i(x_i) + \theta_i(x_i) + \tau_(x_{i-1}, x_i) \tag{11}$$

is a sufficient condition for optimality.

In effect, we solve the LP of equation (10) in two independent procedures, each solving the one-directional subproblem in (6), and either one of these subroutines is sufficient to construct a primal optimal solution. This redundancy is important, though, because the resulting reduced cost

$$
\begin{aligned}
2R_i(x_i, x_{i+1}) = {} & 2\tau(x_i, x_{i+1}) + \theta_i(x_i) + \theta_{i+1}(x_{i+1}) \\
& + (\alpha_i(x_i) - \alpha_{i+1}(x_{i+1})) + (\beta_{i+1}(x_{i+1}) - \beta_i(x_i)).
\end{aligned}
\tag{12}
$$

incorporates global information from both directions in the chain. In table 2 we show that column generation with (12) is fastest, which is not obvious, given the extra overhead of computing the $\beta$ messages. This is the reduced cost that we use in the following discussion and experiments, unless explicitly stated otherwise.

## 4 Column Generation Algorithm

We present an algorithm for exact MAP inference that in practice is usually faster than traditional message passing. Like all column generation algorithms, our technique requires components for three tasks: choosing the initial set of variables in the restricted LP, solving the restricted LP, and finding variables with positive reduced cost. When no variable of positive reduced cost exists, the current solution to the restricted problem is optimal because we have a primal-feasible, dual-feasible pair.

Pseudocode for our algorithm is provided in Figure 1. In the following description, many concepts will be explained in terms of nodes, despite our LP being defined over edges. The edge quantities can be defined in terms of node quantities, such as the $\alpha$ and $\beta$ messages, and it is more efficient to store these than the quadratically-many edge quantities.

### 4.1 Initialization

To initialize the LP, we first define a restricted domain for each node in the graphical model consisting of only $x_i^L = \operatorname{argmax} \theta_i(x_i)$. Other initialization strategies, such as adding the high-scoring transitions, or the $k$ best $x_i$, are also valid. Next, we include in the initial restricted LP all the indicator variables $\mu_i(x_i^L, x_{i+1}^L)$ corresponding to these size-one domains. Solving the initial restricted LP is very efficient, since all nodes have only one valid setting, and no maximization is needed when passing messages.

## 4.2 Warm-Starting the Restricted LP

Updating all messages using the max-product rules of equations (7) and (11) is a valid way to solve the restricted LP, but it doesn't leverage the messages that were optimal for previous calls to the problem. In practice, the restricted domains of every node are not updated at every iteration, and hence many of the previous messages may still appear in a dual-optimal setting of the current restricted problem. As usual, solving the restricted LP, can be decomposed into independently solving each of the one-directional LPs, and thus we update $\alpha$ independently of $\beta$.

To construct a primal setting from either the $\alpha$ or $\beta$ messages, we employ the standard technique of back-tracing the argmaxes used in their update equations. In some regions of the chain, we can avoid updating messages because we can guarantee that the proposed message updates would yield the same maximization and thus the same primal setting. Simple rules include, for example, avoiding updating $\alpha$ to the left of the first updated domain and to avoid updating $\alpha_i(*)$ if $|D_{i-1}| = 1$, since maximization over $|D_{i-1}|$ is trivial. Furthermore, to the right of the the last updated domain, if we compute new messages $\alpha_i'(*)$ and find that the argmax at the current MAP assignment $x_i^*$ doesn't change, we can revert to the previous $\alpha_i(*)$ and terminate message passing. An analogous statement can be made about the $\beta$ variables.

When solving the restricted LP, some constraints are trivially satisfied because they only involve variables that are implicitly set to zero, and hence the corresponding dual variables can be set arbitrarily. To prevent extraneous un-generated variables from having a high reduced cost, we choose duals by guessing values that should be feasible in the unrestricted LP, with a smaller computational cost than solving the unrestricted LP directly. We employ the same update equation used for the in-domain messages in (7) and (11), and maximize over the restricted domain of the variable's neighbor. In our experiments, over 90% of the restricted domains were of size 1, so this dependence on the size of the neighbor domain was not a computational bottleneck in practice, and still allowed the reduced-cost oracle to consider five or less candidate edges in each iteration in more than 86% of the calls.

## 4.3 Reduced-Cost Oracle

Exhaustively searching the chain for variables of positive reduced cost by iterating over all settings of all edges would be as expensive as exact max-product message-passing. However, our oracle search strategy is efficient because it prunes these away using precomputed bounds on the transitions.

First we decompose equation (12) as follows

$$2R_i(x_i, x_{i+1}) = 2\tau(x_i, x_{i+1}) + S_i^+(x_i) + S_i^-(x_{i+1}) \tag{13}$$

where $S_i^+(x_i) = \theta_i(x_i) + \alpha_i(x_i) - \beta_i(x_i)$ and $S_i^-(x_{i+1}) = \theta_{i+1}(x_{i+1}) - \alpha_{i+1}(x_{i+1}) + \beta_{i+1}(x_{i+1})$.

If in practice, most settings for each edge have negative reduced cost, we can efficiently find candidate settings by first upper-bounding $S_i^+(x_i) + 2\tau(x_i, x_{i+1})$, finding all possible values $x_{i+1}$ that could yield a positive reduced cost, and then doing the reverse. Finally, we search over the much smaller set of candidates for $x_i$ and $x_{i+1}$. This strategy is described in Figure 1.

After the first round of column generation, if $R_i(x_i, x_{i+1})$ hasn't changed for every $x_i, x_{i+1}$, then no variables of positive reduced cost can exist because they would have been added in the previous iteration, and we can skip the oracle. This condition can be checked while passing messages.

Lastly, a final pruning strategy is that if there are settings $x_i, x_i'$ such that

$$\theta_i(x_i) + \min_{x_{i-1}} \tau(x_{i-1}, x_i) + \min_{x_{i+1}} \tau(x_i, x_{i+1}) > \theta_i(x_i') + \max_{x_{i-1}} \tau(x_{i-1}, x_i') + \max_{x_{i+1}} \tau(x_i', x_{i+1}), \tag{14}$$

then we know with certainty that setting $x_i'$ is suboptimal. This helps prune the oracle's search space efficiently because the above maxima and minima are data-independent offline computations. We can do so by first linearly searching through the labels for a node for the one with highest local score and then using precomputed bounds on the transition scores to linearly discard states whose upper bound on the score is smaller than the lower bound of the best state.

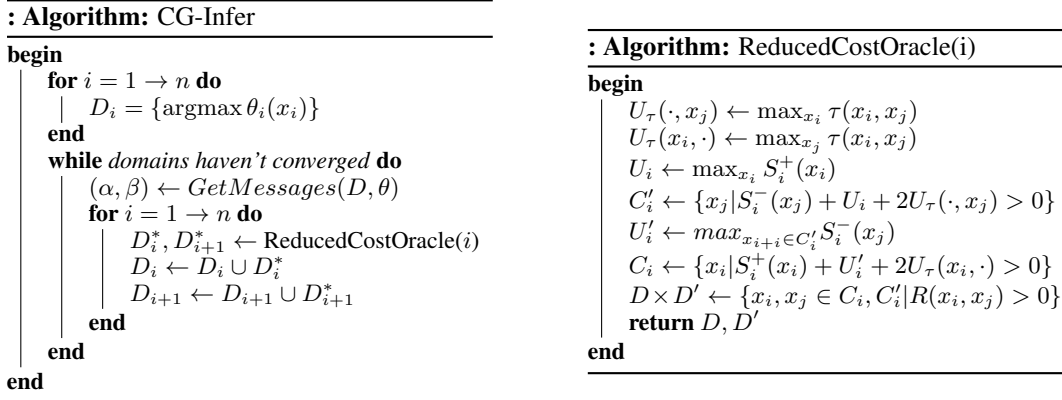

Figure 1: Column Generation Algorithm and Pruning Strategy for Reduced Cost Oracle

## 5 Extensions of the Algorithm

The column generation algorithm is fairly general, and can be easily extended to allow for many interesting use cases. In section 7 we provide experiments supporting the usefulness of these extensions, and they are described in more detail in appendix A.

First of all, our algorithm generalizes easily to MAP inference in trees by using a similar structure but a different reduced cost expression that considers messages flowing in both directions across each edge (appendix A.1). The reduced cost oracle can also be used to compute the duality gap of an approximate solution. This allows early stopping of our algorithm if the gap is small and also provides analysis of the sub-optimality of the output of beam search (appendix A.2). Furthermore, margin violation queries when doing structured SVM training with a 0/1 loss can be done efficiently using a small modification of our algorithm, in which we also add variables of small negative reduced cost and do 2-best inference within the restricted domains (appendix A.3). Lastly, regularizing the transition weights more strongly allows one to train models that will decode more quickly (appendix A.4). Most standard inference algorithms, such as Viterbi, do not have this behavior where the inference time is affected by the actual model scores. By coupling inference and learning, practitioners have more freedom to trade off test-time speed vs. accuracy.

## 6 Related Work

Column generation has been employed as a way of dramatically speeding up MAP inference problems in Riedel et al [10], which applies it directly to the LP relaxation for dependency parsing with grandparent edges.

There has been substantial prior work on improving the speed of max-product inference in chains by pruning the search process. CarpeDiem [11] relies on an an expression similar to the oriented, left-to-right reduced cost equation of (8), also with a similar pruning strategy to the one described in section 4.3. Following up, Kaji et al. [12] presented a staggered decoding strategy that similarly attempts to bound the best achievable score using uninstantiated domains, but only used local scores when searching for new candidates. The dual variables obtained in earlier runs were then used to warm-start the inference in later runs, similarly to what is done in section 4.2. Their techniques obtained similar speed-ups as ours over Viterbi inference. However, their algorithms do not provide extensions to inference in trees, a margin-violation oracle, and approximate inference using a duality gap. Furthermore, Kaji et al. use data-dependent transition scores. This may improve our performance as well, if the transition scores are more sharply peaked. Similarly, Raphael [13] also presents a staggered decoding strategy, but does so in a way that applies to many dynamic programming algorithms.

The strategy of preprocessing data-independent factors to speed up max-product has been previously explored by McAuley and Caetano [14], who showed that if the transition weights are large, savings can be obtained by sorting them offline. Our contributions, on the other hand, are more effective

when the transitions are small. The same authors have also explored strategies to reduce the worst-case complexity of message-passing by exploiting faster matrix multiplication algorithms [15].

Alternative methods of leveraging the interplay between fast dynamic programming algorithms and higher-level LP techniques have been explored elsewhere. For example, in dual decomposition [16], inference in joint models is reduced to repeated inference in independent models. Tree block-coordinate descent performs approximate inference in loopy models using exact inference in trees as a subroutine [1]. Column generation is cutting planes in the dual, and cutting planes have been used successfully in various machine learning contexts. See, for example, Sontag et al [17] and Riedel et al [18].

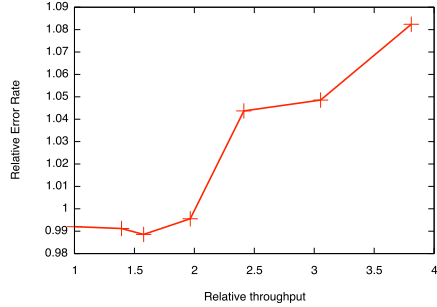

Figure 2: Training-time manipulation of accuracy vs. test throughput for our algorithm

There is a mapping between dynamic programs and shortest path problems [19]. Our reduced cost is an estimate of the desirability of an edge setting, and thus our algorithm is heuristic search in the space of edge settings. With dual feasibility, this heuristic is consistent, and thus our algorithm is iteratively constructing a heuristic such that it can perform $A^*$ search for the final restricted LP [20].

# 7 Experiments

We compare the performance of column generation with exact and approximate inference on Wall Street Journal [21] part-of-speech (POS) tagging and joint POS tagging and named-entity-recognition (POS/NER). The output variable domain size is 45 for POS and 360 for POS/NER. The test set contains 5463 sentences. The POS model was trained with a 0/1 loss structured SVM and the POS/NER model was trained using SampleRank [22].

Table 1 compares the inference times and accuracies of column generation (CG), Viterbi, Viterbi with the final pruning technique described in section 4.3 (Viterbi+P), CG with duality gap termination condition 0.15% (CG+DG), and beam search. For POS, CG, is more than twice as fast as Viterbi, with speed comparable to a beam of size 3. Whereas CG is exact, Beam-3 loses 1.6% accuracy. Exact inference in the model obtains a tagging accuracy of 95.3%.

For joint POS and NER tagging, the speedups are even more dramatic. We observe a 13x speedup over Viterbi and are comparable in speed with a beam of size 7, while being exact. As in POS, CG-DG provides a mild speedup.

Over 90% of tokens in the POS task had a domain of size one, and over 99% had a domain of size 3 or smaller. Column generation always finished in at most three iterations, and 22% of the time it terminated after one. 86% of the time, the reduced-cost oracle iterated over at most 5 candidate edge settings, which is a significant reduction from the worst-case behavior of $45^2$. The pruning strategy in Viterbi+P manages to restrict the number of possible labels for each token to at most 5 for over 65% of the tokens, and prunes the size of each domain by half over 95% of the time.

Table 2.A presents results for a 0/1 loss oracle described in section 5. Baselines are a standard Viterbi 2-best search[1] and Viterbi 2-best with the pruning technique of 4.3 (Viterbi+P). CG outperforms Viterbi 2-best on both POS and POS/NER. Though Viterbi+P presents an effective speedup, we are still 19x faster on POS/NER. In terms of absolute throughput, POS/NER is faster than POS because the POS/NER model wasn't trained with a regularized structured SVM, and thus there are fewer margin violations. Our 0/1 oracle is quite efficient when determining that there isn't a margin violation, but requires extra work when required to actually produce the 2-best setting.

Table 2.B shows column generation with two other reduced-cost formulations on the same POS tagging task. CG-$\alpha$ uses the reduced-cost from equation (8) while CG-$\alpha+\theta_{i+1}$ uses the reduced-cost from equation (9). The full CG is clearly beneficial, despite requiring computation of $\beta$.

| Algorithm | % Exact | Sent./sec. |
|---|---|---|
| Viterbi | 100 | 3144.6 |
| Viterbi+P | 100 | 4515.3 |
| CG | 100 | 8227.6 |
| CG-DG | 98.9 | 9355.6 |
| Beam-1 | 57.7 | 12117.6 |
| Beam-2 | 92.6 | 7519.3 |
| Beam-3 | 98.4 | 6802.5 |
| Beam-4 | 99.5 | 5731.2 |

| Algorithm | % Exact | Sent./sec. |
|---|---|---|
| Viterbi | 100 | 56.9 |
| Viterbi+P | 100 | 498.9 |
| CG | 100 | 779.9 |
| CG-DG | 98.4 | 804 |
| Beam-1 | 66.6 | 3717.0 |
| Beam-5 | 98.5 | 994.97 |
| Beam-7 | 99.2 | 772.8 |
| Beam-10 | 99.5 | 575.1 |

Table 1: Comparing inference time and exactness of Column Generation (CG), Viterbi, Viterbi with the final pruning technique of section 4.3 (Viterbi+P), and CG with duality gap termination condition 0.15%(CG+DG), and beam search on POS tagging (left) and joint POS/NER (right).

| Method | POS Sent./sec. | POS/NER Sent./sec. |
|---|---|---|
| CG | 85.0 | 299.9 |
| Viterbi 2-best | 56.0 | .06 |
| Viterbi+P 2-best | 119.6 | 11.7 |

| Reduced Cost | POS Sent./sec. |
|---|---|
| CG | 8227.6 |
| CG-$\alpha$ | 5125.8 |
| CG-$\alpha+\theta_{i+1}$ | 4532.1 |

Table 2: (A) the speedups for a 0/1 loss oracle (B) comparing reduced cost formulations

In Figure 2, we explore the ability to manipulate training time regularization to trade off test accuracy and test speed, as discussed in section 5. We train a structured SVM with $L_2$ regularization (coefficient $0.1$) the emission weights, and vary the $L_2$ coefficient on the transition weights from $0.1$ to $10$. A 4x gain in speed can be obtained at the expense of an 8% relative decrease in accuracy.

# 8 Conclusions and future work

In this paper we presented an efficient family of algorithms based on column generation for MAP inference in chains and trees. This algorithm exploits the fact that inference can often rule out many possible values, and we can efficiently expand the set of values on the fly. Depending on the parameter settings it can be twice as fast as Viterbi in WSJ POS tagging and 13x faster in a joint POS-NER task.

One avenue of further work is to extend the bounding strategies in this algorithm for inference in cluster graphs or junction trees, allowing faster inference in higher-order chains or even loopy graphical models. The connection between inference and learning shown in section 5 also bears further study, since it would be helpful to have more prescriptive advice for regularization strategies to achieve certain desired accuracy/time tradeoffs.

# Acknowledgments

This work was supported in part by the Center for Intelligent Information Retrieval. The University of Massachusetts gratefully acknowledges the support of Defense Advanced Research Projects Agency (DARPA) Machine Reading Program under Air Force Research Laboratory (AFRL) prime contract no. FA8750-09-C-0181, in part by IARPA via DoI/NBC contract #D11PC20152, in part by Army prime contract number W911NF-07-1-0216 and University of Pennsylvania subaward number 103-548106 , and in part by UPenn NSF medium IIS-0803847. Any opinions, findings and conclusions or recommendations expressed in this material are the authors' and do not necessarily reflect those of the sponsor. The U.S. Government is authorized to reproduce and distribute reprint for Governmental purposes notwithstanding any copyright annotation thereon

## Footnotes

\*The first two authors contributed equally to this paper.

[1]Implemented by replacing all maximizations in the viterbi code with two-best maximizations.

# References

[1] David Sontag and Tommi Jaakkola. Tree block coordinate descent for MAP in graphical models. In *Proceedings of the Twelfth International Conference on Artificial Intelligence and Statistics (AI-STATS)*, volume 8, pages 544–551. JMLR: W&CP, 2009.

[2] C. Pal, C. Sutton, and A. McCallum. Sparse forward-backward using minimum divergence beams for fast training of conditional random fields. In *Acoustics, Speech and Signal Processing, 2006. ICASSP 2006 Proceedings. 2006 IEEE International Conference on*, volume 5, pages V–V. IEEE, 2006.

[3] A. Kulesza, F. Pereira, et al. Structured learning with approximate inference. *Advances in neural information processing systems*, 20:785–792, 2007.

[4] L. Shen, G. Satta, and A. Joshi. Guided learning for bidirectional sequence classification. In *Annual Meeting-Association for Computational Linguistics*, volume 45, page 760, 2007.

[5] D. Tarlow, D. Batra, P. Kohli, and V. Kolmogorov. Dynamic tree block coordinate ascent. In *ICML*, pages 113–120, 2011.

[6] C. Yanover, T. Meltzer, and Y. Weiss. Linear programming relaxations and belief propagation–an empirical study. *The Journal of Machine Learning Research*, 7:1887–1907, 2006.

[7] M. Lubbecke and J. Desrosiers. Selected topics in column generation. *Operations Research*, 53:1007–1023, 2004.

[8] D. Bertsimas and J. Tsitsiklis. *Introduction to Linear Optimization*. Athena Scientific, 1997.

[9] M.J. Wainwright and M.I. Jordan. Graphical models, exponential families, and variational inference. *Foundations and Trends in Machine Learning*, 1(1-2):1–305, 2008.

[10] S. Riedel, D. Smith, and A. McCallum. Parse, price and cutdelayed column and row generation for graph based parsers. *Proceedings of the Conference on Empirical methods in natural language processing (EMNLP '12)*, 2012.

[11] R. Esposito and D.P. Radicioni. Carpediem: an algorithm for the fast evaluation of SSL classifiers. In *Proceedings of the 24th international conference on Machine learning*, pages 257–264. ACM, 2007.

[12] N. Kaji, Y. Fujiwara, N. Yoshinaga, and M. Kitsuregawa. Efficient staggered decoding for sequence labeling. In *Proceedings of the 48th Annual Meeting of the Association for Computational Linguistics*, pages 485–494. Association for Computational Linguistics, 2010.

[13] C. Raphael. Coarse-to-fine dynamic programming. *Pattern Analysis and Machine Intelligence, IEEE Transactions on*, 23(12):1379–1390, 2001.

[14] J. McAuley and T. Caetano. Exploiting data-independence for fast belief-propagation. In *International Conference on Machine Learning 2010*, volume 767, page 774, 2010.

[15] J.J. McAuley and T.S. Caetano. Faster algorithms for max-product message-passing. *Journal of Machine Learning Research*, 12:1349–1388, 2011.

[16] A.M. Rush, D. Sontag, M. Collins, and T. Jaakkola. On dual decomposition and linear programming relaxations for natural language processing. In *Proceedings of the 2010 Conference on Empirical Methods in Natural Language Processing*, pages 1–11. Association for Computational Linguistics, 2010.

[17] D. Sontag and T. Jaakkola. New outer bounds on the marginal polytope. In *Advances in Neural Information Processing Systems*, 2007.

[18] S. Riedel. Improving the accuracy and efficiency of MAP inference for Markov logic. *Proceedings of UAI 2008*, pages 468–475, 2008.

[19] R. Kipp Martin, Ronald L. Rardin, and Brian A. Campbell. Polyhedral characterization of discrete dynamic programming. *Operations Research*, 38(1):pp. 127–138, 1990.

[20] R.K. Ahuja, T.L. Magnanti, J.B. Orlin, and K. Weihe. Network flows: theory, algorithms and applications. *ZOR-Methods and Models of Operations Research*, 41(3):252–254, 1995.

[21] M.P. Marcus, M.A. Marcinkiewicz, and B. Santorini. Building a large annotated corpus of english: The penn treebank. *Computational linguistics*, 19(2):313–330, 1993.

[22] M. Wick, K. Rohanimanesh, K. Bellare, A. Culotta, and A. McCallum. SampleRank: training factor graphs with atomic gradients. In *Proceedings of ICML*, 2011.

